# Modeling Applications with the Focused Gamma Net

Jose C. Principe, Bert de Vries, Jyh-Ming Kuo and Pedro Guedes de Oliveira*

Department of Electrical Engineering
University of Florida, CSE 447
Gainesville, FL 32611
principe@synapse.ee.ufl.edu

*Departamento Eletronica/INESC
Universidade de Aveiro
Aveiro, Portugal

## Abstract

The focused gamma network is proposed as one of the possible implementations of the gamma neural model. The focused gamma network is compared with the focused backpropagation network and TDNN for a time series prediction problem, and with ADALINE in a system identification problem.

## 1    INTRODUCTION

At NIPS-90 we introduced the *gamma neural model*, a real time neural net for temporal processing (de Vries and Principe, 1991). This model is characterized by a neural short term memory mechanism, the gamma memory structure, which is implemented as a tapped delay line of adaptive dispersive elements. The gamma model seems to provide an integrative framework to study the neural processing of time varying patterns (de Vries and Principe, 1992). In fact both the memory by delays as implemented in TDNN (Lang et al, 1990) and memory by local feedback (self-recurrent loops) as proposed by Jordan (1986), and Elman (1990) are special cases of the gamma memory structure. The preprocessor utilized in Tank's and Hopfield concentration in time (CIT) network (Tank and Hopfield, 1989) can be shown to be very similar to the dispersive structure utilized in the gamma memory (deVries, 1991). We studied the gamma memory as an independent adaptive filter structure (Principe et al, 1992), and concluded that it is a special case of a class of IIR (infinite impulse response) adaptive filters, which we called the generalized feedforward structures. For these structures, the well known Wiener-Hopf solution to find the optimal filter weights can be analytically computed. One of the advantages of the gamma memory as an adaptive filter is that, although being a recursive structure, stability is easily ensured. Moreover, the LMS algorithm can be easily

extended to adapt all the filter weights, including the parameter that controls the depth of memory, with the same complexity as the conventional LMS algorithm (i.e. the algorithm complexity is linear in the number of weights). Therefore, we achieved a theoretical framework to study memory mechanisms in neural networks.

In this paper we compare the gamma neural model with other well established neural networks that process time varying signals. Therefore the first step is to establish a topology for the gamma model. To make the comparison easier with respect to TDNN and Jordan's networks, we will present our results based on the focused gamma network. The focused gamma network is a multilayer feedforward structure with a gamma memory plane in the first layer (Figure 1). The learning equations for the focused gamma network and its memory characteristics will be addressed in detail. Examples will be presented for prediction of complex biological signals (electroencephalogram-EEG) and chaotic time series, as well as a system identification example.

## 2    THE FOCUSED GAMMA NET

The focused neural architecture was introduced by Mozer (1988) and Stornetta et al (1988). It is characterized by a a two stage topology where the input stage stores traces of the input signal, followed by a nonlinear continuous feedforward mapper network (Figure 1). The gamma memory plane represents the input signal in a time-space plane (spatial dimension M, temporal dimension K). The activations in the memory layer are $I_{ik}(t)$, and the activations in the feedforward network are represented by $x_i(t)$. Therefore the following equations apply respectively for the input memory plane and for the feedforward network,

$$I_0(t) = I_i(t)$$
$$I_{ik}(t) = (1-\mu_i)I_{ik}(t-1) + \mu_i I_{i,k-1}(t-1), \; \text{i=1,...,M; k=1,...,K.} \qquad (1)$$

$$x_i(t) = \sigma\left(\sum_{j<i} w_{ij}x_j(t) + \sum_{j,k} w_{ijk}I_{jk}(t)\right) \qquad \text{, i=1,...,N.} \qquad (2)$$

where $\mu_i$ is an adaptive parameter that controls the depth of memory (Principe et al, 1992), and $w_{ijk}$ are the spatial weights. Notice that the focused gamma network for K=1 is very similar to the focused-backpropagation network of Mozer and Stornetta. Moreover, when $\mu=1$ the gamma memory becomes a tapped delay line which is the configuration utilized in TDNN, with the time-to-space conversion restricted to the first layer (Lang et al, 1990). Notice also that if the nonlinear feedforward mapper is restricted to one layer of linear elements, and $\mu=1$, the focused gamma memory becomes the adaptive linear combiner - ADALINE (Widrow et al,1960).

In order to better understand the computational properties of the gamma memory we defined two parameters, the mean memory depth D and memory resolution R as

$$D = \frac{K}{\mu} \qquad\qquad R = \frac{K}{D} = \mu \qquad\qquad (3)$$

(de Vries, 1991). Memory depth measures how far into the past the signal conveys information for the processing task, while resolution quantifies the temporal proximity of the memory traces.

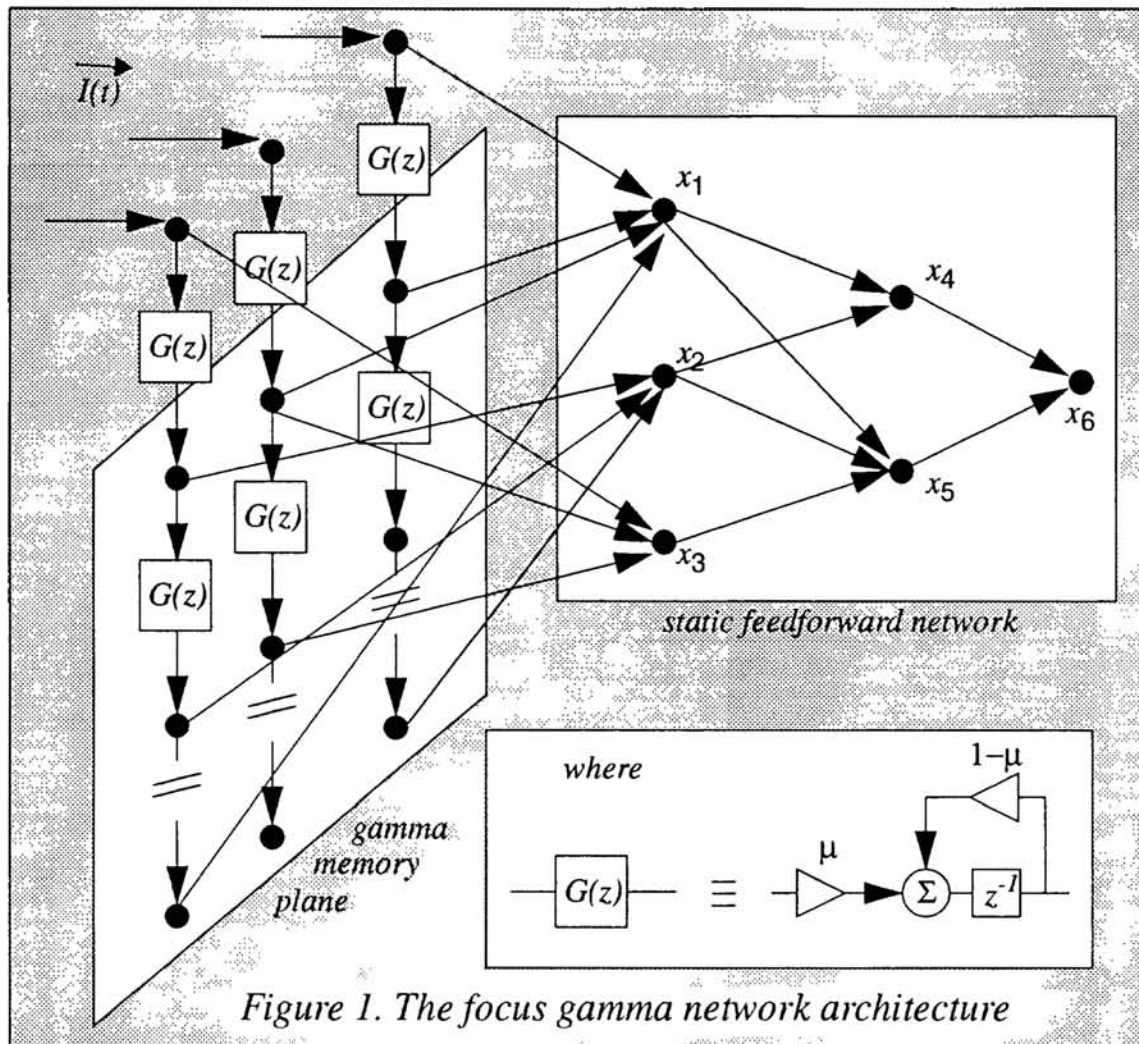

*Figure 1. The focus gamma network architecture*

The important aspect in the gamma memory formalism is that $\mu$, which controls both the memory resolution and depth, is an adaptive parameter that is learned from the signal according to the optimization of a performance measure. Therefore the focused gamma network always works with the optimal memory depth/ resolution for the processing problem. The gamma memory is an adaptive recursive structure, and as such can go unstable during adaptation. But due to the local feedback nature of $G(z)$, stability is easily ensured by keeping $0<\mu<2$.

The focused gamma network is a recurrent neural model, but due to the topology selected, the spatial weights can be learned using regular backpropagation (Rumelhart et al, 1986). However for the adaptation of $\mu$, a recurrent learning procedure is necessary. Since most of the times the order of the gamma memory is small, we recommend adapting $\mu$ with direct differentiation using the real time recurrent learning (RTRL) algorithm (Williams and Zipzer,1989), which when applied to the gamma memory yields,

$$\frac{\partial}{\partial\mu_i}E(t) = \sum_m \frac{\partial E}{\partial x_m(t)} x \left( \frac{\partial x_m(t)}{\partial I_{ik}(t)} \, x \, \frac{\partial I_{ik}(t)}{\partial \mu_i} \right)$$

$$= -\sum_m e_m(t)\,\sigma'\,[net_m(t)]\sum_k w_{mik}\alpha_i^k(t)$$

where by definition $\alpha_i^k(t) = \frac{\partial}{\partial\mu_i}I_{ik}(t)$, and

$$\frac{\partial}{\partial\mu_i}I_{ik}(t) = (1-\mu_i)\alpha_i^k(t-1) + \mu_i\alpha_i^{k-1}(t-1) + [I_{i,k-1}(t-1) - I_{i,k}(t-1)]$$

However, backpropagation through time (BPTT) (Werbos, 1990) can also be utilized, and will be more efficient when the temporal patterns are short.

## 3     EXPERIMENTAL RESULTS

The results for prediction that will be presented here utilized the focused gamma network as depicted in Figure 2a, while for the case of system identification, the block diagram is presented in Figure 2b.

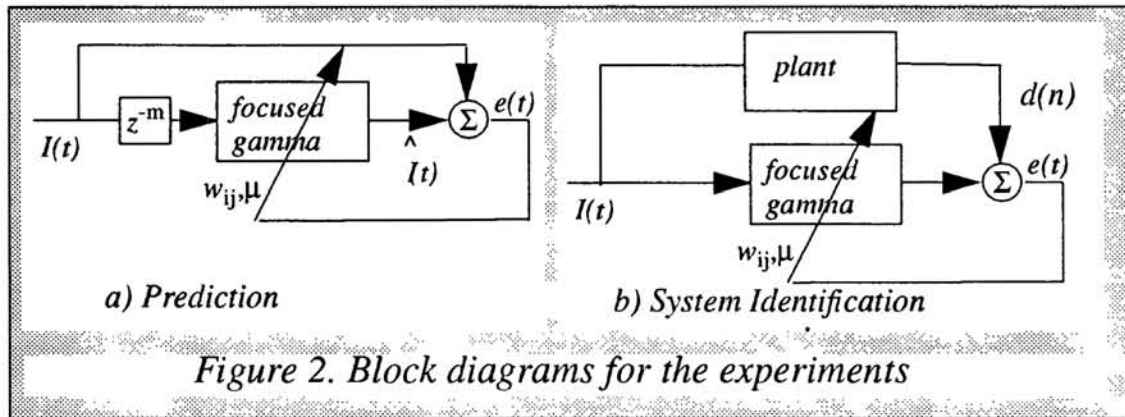

*Figure 2. Block diagrams for the experiments*

Prediction of EEG

We selected an EEG signal segment for our first comparison, because the EEG is notorious for its complexity. The problem was to predict the signal five steps ahead (feedforward prediction). Figure 3 shows a four second segment of sleep stage 2. The topology utilized was K gamma units, a one-hidden layer configuration with 5 units (nonlinear) and one linear output unit. The performance criterion is the mean square error signal. We utilized backpropagation to adapt the spatial weights ($w_{ijk}$), and parametrized $\mu$ between 0 and 1 in steps of 0.1. Figure 3b displays the curves of minimal mean square error versus $\mu$.

One can immediately see that the minimum mean square error is obtained for values of $\mu$ different from one, therefore for the same memory order the gamma memory outperforms the tapped delay line as utilized in TDNN (which once again is equivalent to the gamma memory for $\mu=1$). For the case of the EEG it seems that the advantage of the gamma memory diminishes when the order of the memory is

increased. However, the case of K=2, μ=0.6 produces equivalent performance of a TDNN with 4 memory taps (K=4). Since in experimental conditions there is always noise, experience has shown that the fewer number of adaptive parameters yield better signal fitting and simplifies training, so the focused gamma network is preferable.

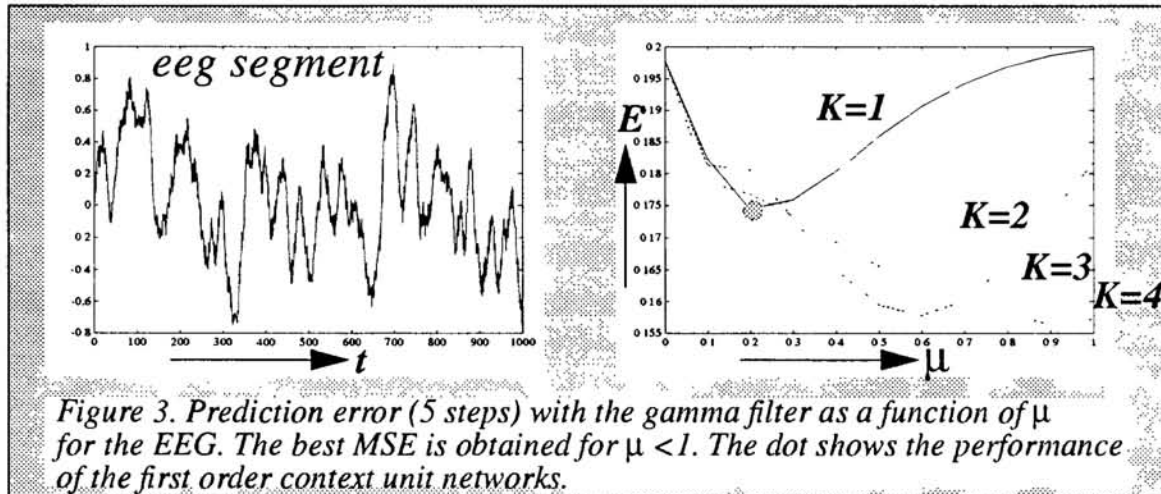

*Figure 3. Prediction error (5 steps) with the gamma filter as a function of μ for the EEG. The best MSE is obtained for μ <1. The dot shows the performance of the first order context unit networks.*

Notice also that the case of networks with first order context unit is obtained for K=1, so even if the time constant is chosen right (μ=0.2 in this case), the performance can be improved if higher order memory kernels are utilized. It is also interesting to note that the optimal memory depth for the EEG prediction problem seems to be around 4, as this is the value of K/μ $_{optimal}$. The information regarding the "optimal memory depth" is not obtainable with conventional models.

### Prediction of Mackey-Glass time series

The Mackey-Glass system is a delay differential equation that becomes chaotic for some values of the parameters and delays (Mackey-Glass, 1977). The results that will be presented here regard the Mackey-Glass system with delay D=30. The time series was generated using a fourth order Runge-Kutta algorithm. The table in Figure 4 shows the performance of TDNN and the focused gamma network with the same number of free parameters. The number of hidden units was kept the same in both networks, but TDNN utilized 5 input units, while the focused gamma network had 4 input units, and the adaptive memory depth parameter μ. The two systems were trained with the same number of samples, and training epochs. For TDNN this was the value that gave the best training when cross validation was utilized (the error in the test set increased after 100 epochs). For this example μ was adapted on-line using RTRL, with the initial value set at μ=1, and with the same step size as for the spatial weights. As the Table shows, the MSE in the training for the gamma network is substantially lower than for TDNN. Figure 4 shows the behavior of μ during the training epochs. It is interesting to see that the value of μ changes during training and settles around a value of 0.92. In terms of learning curve (the MSE as a function of epoch) notice that there is an intersection of the learning curves for the TDNN and gamma network around epoch 42 when the value of μ=1, as we could expect from our analysis. The gamma network starts outperforming TDNN when the correct value of μ is approached.

This example shows that μ can be learned on line, and that once again having the freedom to select the right value of memory depth helps in terms of prediction performance. For both these cases the required memory depth is relatively shallow, what we can expect since a chaotic time series has positive Lyapunov exponents, so the important information to predict the next point is in the short-term past. The same argument applies to the EEG, that can also be modeled as a chaotic time series (Lo and Principe, 1989). Cases where the long-term past is important for the task should enhance the advantage of the gamma memory.

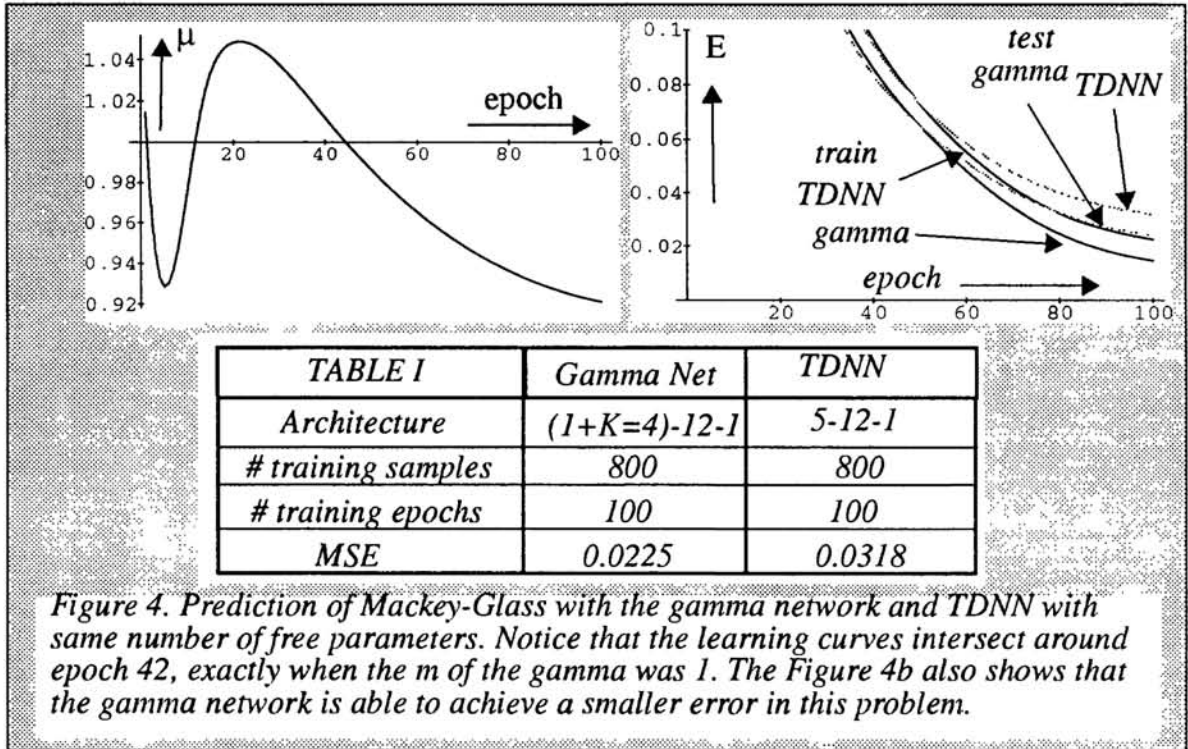

| TABLE I | Gamma Net | TDNN |
|---|---|---|
| Architecture | (1+K=4)-12-1 | 5-12-1 |
| # training samples | 800 | 800 |
| # training epochs | 100 | 100 |
| MSE | 0.0225 | 0.0318 |

Figure 4. Prediction of Mackey-Glass with the gamma network and TDNN with same number of free parameters. Notice that the learning curves intersect around epoch 42, exactly when the m of the gamma was 1. The Figure 4b also shows that the gamma network is able to achieve a smaller error in this problem.

## Linear System Identification

The last example is the identification of a third order linear lowpass elliptic transfer function with poles and zeros, given by

$$H(z) = \frac{1 - 0.8731z^{-1} - 0.8731z^{-2} + z^{-3}}{1 - 2.8653z^{-1} + 2.7505z^{-2} - 0.8843z^{-3}}$$

The cutoff frequency of this filter was selected such that the impulse response was long, effectively creating the need for a deep memory for good identification. For this case the focused gamma network was reduced to an ADALINE(μ) (deVries et al, 1991), i.e. the feedforward mapper was a one layer linear network. The block diagram of Figure 2b was utilized to train the gamma network, and I(t) was chosen to be white gaussian noise. Figure 5 shows the MSE as a function of μ for gamma memory orders up to k=3. Notice that the information gained from the Figure 5 agrees with our speculations. The optimal value of the memory is K/μ ~ 17 samples. For this value the third order ADALINE performs very poorly because there is not enough information in 3 delays to identify the transfer function with small error. The gamma memory, on the other hand can choose μ small to encompass the required

length, even for a third order memory. The price paid is reduced resolution, but the performance is still much better than the ADALINE of the same order (a factor of 10 improvement).

## 4    CONCLUSIONS

In this paper we propose a specific topology of the gamma neural model, the focused gamma network. Several important neural networks become special cases of the focused gamma network. This allowed us to compare the advantages of having a more versatile memory structure than any of the networks under comparison.

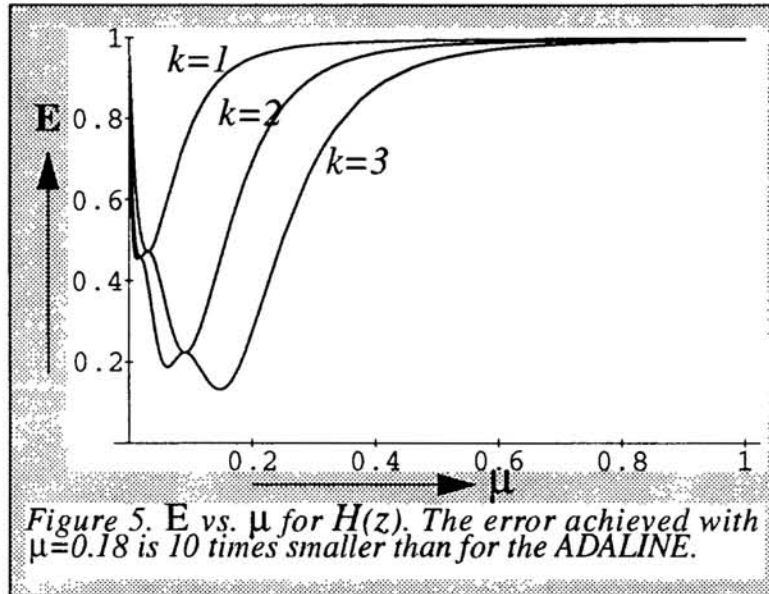

*Figure 5. $E$ vs. $\mu$ for $H(z)$. The error achieved with $\mu=0.18$ is 10 times smaller than for the ADALINE.*

The conclusion is that the gamma memory is computationally more powerful than fixed delays or first order context units. The major advantage is that the gamma model formalism allows the memory depth to be optimally set for the problem at hand. In the case of the chaotic time series, where the information to predict the future is concentrated in the neighborhood of the present sample, the gamma memory selected the most appropriate value, but its performance is similar to TDNN. However, in cases where the required depth of memory is much larger than the size of the tapped delay line, the gamma memory outperforms the fixed depth topologies with the same number of free parameters.

The price paid for this optimal performance is insignificant. As a matter of fact, $\mu$ can be adapted in real-time with RTRL (or BPTT), and since it is a single global parameter the complexity of the algorithm is still $O(K)$ with RTRL. The other possible problem, instability, is easily controlled by requiring that the value of $\mu$ be limited to $0<\mu<2$.

The focused gamma memory is just one of the possible neural networks that can be implemented with the gamma model. The use of gamma memory planes in the hidden or output processing elements will enhance the computational power of the neural network. Notice that in these cases the short term mechanism is not only utilized to store information of the signal past, but will also be utilized to store the past values of the neural states. We can expect great savings in terms of network size with these

other structures, mainly in cases where the information of the long-term past is important for the processing task.

## Acknowledgments

This work has been partially supported by NSF grant DDM-8914084.

## References

De Vries B. and Principe J.C. (1991). A Theory for Neural Nets with Time Delays. In Lippmann R., Moody J., and Touretzky D. (eds.), *NIPS90 proceedings*, San Mateo, CA, Morgan Kaufmann.

DeVries B., Principe J., Oliveira P. (1991). Adaline with Adaptive Recursive Memory, *Proc. IEEE Work. Neural Nets for Sig. Proc.*, Princeton, 101-110, IEEE Press.

DeVries and Principe, (1992). The gamma neural net - A new model for temporal processing. Accepted for publication, *Neural Networks*.

DeVries B.(1991). Temporal Processing with Neural Networks- The Development of the Gamma Model, Ph.D. Dissertation, *University of Florida*.

Elman, (1988). Finding structure in time. *CRL technical report 8801*, 1988.

Jordan, (1986). Attractor dynamics and parallelism in a connectionist sequential machine. *Proc. Cognitive Science* 1986.

Lang et. al. (1990). A time-delay neural network architecture for isolated word recognition. *Neural Networks, vol.3 (1)*, 1990.

Lo P.C. and Principe, J.C. (1989). Dimensionality analysis of EEG Segments: experimental considerations, *Proc. IJCNN 89*, vol I, 693-698.

Mackey D., Glass L. (1977). Oscillation and Chaos in Physiological Control Systems, *Science 197*, 287.

Mozer M.C. (1989). A Focused Backpropagation Algorithm for Temporal Pattern Recognition. *Complex Systems* 3, 349-381.

Principe J.C., De Vries B., Oliveira, P.(1992). The Gamma Filter - A New Class of Adaptive IIR Filters with Restricted Feedback. Accepted for publication in *IEEE Transactions on Signal Processing*.

Rumelhart D.E., Hinton G.E. and Williams R.J. (1986). Learning Internal Representations by Error Back-propagation. In Rumelhart D.E., McClelland J.L. (eds.) , *Parallel Distributed Processing*, vol. 1, ch. 8, MIT Press.

Stornetta W.S., Hogg T. and Huberman B.A. (1988). A Dynamical Approach to Temporal Pattern Processing. In Anderson D.Z. (ed.), *Neural Information Processing Systems*, 750-759.

Tank and Hopfield, (1987). Concentrating information in time: analog neural networks with applications to speech recognition problems. *1st int. conf. on neural networks*, IEEE, 1987.

Werbos P. (1990). Backpropagation through time:what it does and how to do it., *Proc. IEEE*, vol 78, no 10, 1550-1560.

Widrow B., Hoff M. (1960). Adaptive Switching Circuits, *IRE Wescon Conf. Rep.*, pt 4.

Williams J., and Zipzer D (1989). A learning algorithm for continually tunning fiully recurrent neural networks, *Neural Computation*, vol 1 no 2, pp 270-281, MIT Press.